# Reinforcement Learning in Continuous Action Spaces through Sequential Monte Carlo Methods

**Alessandro Lazaric**    **Marcello Restelli**    **Andrea Bonarini**
Department of Electronics and Information
Politecnico di Milano
piazza Leonardo da Vinci 32, I-20133 Milan, Italy
{bonarini,lazaric,restelli}@elet.polimi.it

## Abstract

Learning in real-world domains often requires to deal with continuous state and action spaces. Although many solutions have been proposed to apply Reinforcement Learning algorithms to continuous state problems, the same techniques can be hardly extended to continuous action spaces, where, besides the computation of a good approximation of the value function, a fast method for the identification of the highest-valued action is needed. In this paper, we propose a novel actor-critic approach in which the policy of the actor is estimated through sequential Monte Carlo methods. The importance sampling step is performed on the basis of the values learned by the critic, while the resampling step modifies the actor's policy. The proposed approach has been empirically compared to other learning algorithms into several domains; in this paper, we report results obtained in a control problem consisting of steering a boat across a river.

## 1 Introduction

Most of the research on Reinforcement Learning (RL) [13] has studied solutions to finite Markov Decision Processes (MDPs). On the other hand, learning in real-world environments requires to deal with continuous state and action spaces. While several studies focused on problems with continuous states, little attention has been deserved to tasks involving continuous actions. Although several tasks may be (suboptimally) solved by coarsely discretizing the action variables (for instance using the tile coding approach [11, 12]), a different approach is required for problems in which high-precision control is needed and actions slightly different from the optimal one lead to very low utility values. In fact, since RL algorithms need to experience each available action several times to estimate its utility, using very fine discretizations may be too expensive for the learning process. Some approaches, although using a finite set of target actions, deal with this problem by selecting real-valued actions obtained by interpolation of the available discrete actions on the basis of their utility values [9, 14]. Despite of this capability, the learning performance of these algorithms relies on strong assumptions about the shape of the value function that are not always satisfied in highly non-linear control problems. The *wire fitting* algorithm [2] (later adopted also in [4]) tries to solve this problem by implementing an adaptive interpolation scheme in which a finite set of pairs $\langle action, value \rangle$ is modified in order to better approximate the action value function.

Besides having the capability of selecting any real-valued action, RL algorithms for continuous action problems should be able to efficiently find the greedy action, i.e., the action associated to the highest estimated value. Differently from the finite MDP case, a full search in a continuous action space to find the optimal action is often unfeasible. To overcome this problem, several approaches limit their search over a finite number of points. In order to keep low this number, many algorithms (e.g., tile coding and interpolation-based) need to make (often implicit) assumptions about the shape of the value function. To overcome these difficulties, several approaches have adopted the actor-

critic architecture [7, 10]. The key idea of actor-critic methods is to explicitly represent the policy (stored by the actor) with a memory structure independent of the one used for the value function (stored by the critic). In a given state, the policy followed by the agent is a probability distribution over the action space, usually represented by parametric functions (e.g., Gaussians [6], neural networks [14], fuzzy systems [5]). The role of the critic is, on the basis of the estimated value function, to criticize the actions taken by the actor, which consequently modifies its policy through a stochastic gradient on its parameter space. In this way, starting from a fully exploratory policy, the actor progressively changes its policy so that actions that yield higher utility values are more frequently selected, until the learning process converges to the optimal policy. By explicitly representing the policy, actor-critic approaches can efficiently implement the action selection step even in problems with continuous action spaces.

In this paper, we propose to use a Sequential Monte Carlo (SMC) method [8] to approximate the sequence of probability distributions implemented by the actor, thus obtaining a novel actor-critic algorithm called SMC-learning. Instead of a parametric function, the actor represents its stochastic policy by means of a finite set of random samples (i.e., actions) that, using simple resampling and moving mechanisms, is evolved over time according to the values stored by the critic. Actions are initially drawn from a prior distribution, and then they are resampled according to an *importance sampling* estimate which depends on the utility values learned by the critic. By means of the resampling and moving steps, the set of available actions gets more and more thick around actions with larger utilities, thus encouraging a detailed exploration of the most promising action-space regions, and allowing SMC-learning to find real continuous actions. It is worth pointing out that the main goal here is not an accurate approximation of the action-value function on the whole action space, but to provide an efficient way to converge to the *continuous* optimal policy. The main characteristics of the proposed approach are: the agent may learn to execute any continuous action, the action selection phase and the search for the action with the best estimated value are computationally efficient, no assumption on the shape of the value function is required, the algorithm is model-free, and it may learn to follow also stochastic policies (needed in multi-agent problems).

In the next section, we introduce basic RL notation and briefly discuss issues about learning with continuous actions. Section 3 details the proposed learning approach (SMC-Learning), explaining how SMC methods can be used to learn in continuous action spaces. Experimental results are discussed in Section 4, and Section 5 draws conclusions and contains directions for future research.

## 2 Reinforcement Learning

In reinforcement learning problems, an agent interacts with an unknown environment. At each time step, the agent observes the *state*, takes an *action*, and receives a *reward*. The goal of the agent is to learn a *policy* (i.e., a mapping from states to actions) that maximizes the long-term return. An RL problem can be modeled as a Markov Decision Process (MDP) defined by a quadruple $\langle \mathcal{S}, \mathcal{A}, \mathcal{T}, \mathcal{R}, \gamma \rangle$, where $\mathcal{S}$ is the set of states, $\mathcal{A}(s)$ is the set of actions available in state $s$, $\mathcal{T} : \mathcal{S} \times \mathcal{A} \times \mathcal{S} \to [0, 1]$ is a transition distribution that specifies the probability of observing a certain state after taking a given action in a given state, $\mathcal{R} : \mathcal{S} \times \mathcal{A} \to \Re$ is a reward function that specifies the instantaneous reward when taking a given action in a given state, and $\gamma \in [0, 1)$ is a discount factor. The policy of an agent is characterized by a probability distribution $\pi(a|s)$ that specifies the probability of taking action $a$ in state $s$. The utility of taking action $a$ in state $s$ and following a policy $\pi$ thereafter is formalized by the action-value function $Q^{\pi}(s, a) = E\left[\sum_{t=1}^{\infty} \gamma^{t-1} r_t | s = s_1, a = a_1, \pi\right]$, where $r_1 = \mathcal{R}(s, a)$. RL approaches aim at learning the policy that maximizes the action-value function in each state. The optimal action-value function can be computed by solving the Bellman equation: $Q^*(s, a) = \mathcal{R}(s, a) + \gamma \sum_{s'} \mathcal{T}(s, a, s') \max_{a'} Q^*(s', a')$. The optimal policy can be defined as the greedy action in each state: $\pi^*(a|s)$ is equal to $1/|arg \max_a Q^*(s, a)|$ if $a \in arg \max_a Q^*(s, a)$, and 0 otherwise.

Temporal Difference (TD) algorithms [13] allows the computation of $Q^*(s, a)$ by direct interaction with the environment. Given the tuple $\langle s_t, a_t, r_t, s_{t+1}, a_{t+1} \rangle$ (i.e., the experience performed by the agent), at each step, action values may be estimated by online algorithms, such as SARSA, whose update rule is:

$$Q(s_t, a_t) \leftarrow (1 - \alpha)Q(s_t, a_t) + \alpha u(r_t, a_{t+1}, s_{t+1}), \tag{1}$$

where $\alpha \in [0, 1]$ is a learning rate and $u(r_t, a_{t+1}, s_{t+1}) = r_t + \gamma Q(s_{t+1}, a_{t+1})$ is the target utility.

Although value-function approaches have theoretical guarantees about convergence to the optimal policy and have been proved to be effective in many applications, they have several limitations: algorithms that maximize the value function cannot solve problems whose solutions are stochastic policies (e.g., multi-agent learning problems); small errors in the estimated value of an action may lead to discontinuous changes in the policy [3], thus leading to convergence problems when function approximators are considered. These problems may be overcome by adopting *actor-critic* methods [7] in which the action-value function and the policy are stored into two distinct representations. The actor typically represents the distribution density over the action space through a function $\pi(a|s, \theta)$, whose parameters $\theta$ are updated in the direction of performance improvement, as established by the critic on the basis of its approximation of the value function, which is usually computed through an on-policy TD algorithm.

## 3 SMC-Learning for Continuous Action Spaces

SMC-learning is based on an actor-critic architecture, in which the actor stores and updates, for each state $s$, a density distribution $\pi^t(a|s)$ that specifies the agent's policy at time instant $t$. At the beginning of the learning process, without any prior information about the problem, the actor usually considers a uniform distribution over the action space, thus implementing a fully exploratory policy. As the learning process progresses, the critic collects data for the estimation of the value function (in this paper, the critic estimates the action-value function), and provides the actor with information about which actions are the most promising. On the other hand, the actor changes its policy to improve its performance and to progressively reduce the exploration in order to converge to the optimal deterministic policy. Instead of using parametric functions, in SMC-learning the actor represents its evolving stochastic policy by means of Monte Carlo sampling. The idea is the following: for each state $s$, the set of available actions $\mathcal{A}(s)$ is initialized with $N$ samples drawn from a proposal distribution $\pi^0(a|s)$:

$$\mathcal{A}(s) = \{a_1, a_2, \cdots, a_N\}, \quad a_i \sim \pi^0(a|s).$$

Each sampled action $a_i$ is associated to an importance weight $w_i \in \mathcal{W}(s)$ whose value is initialized to $1/N$, so that the prior density can be approximated as

$$\pi^0(a|s) \simeq \sum_{i=1}^{N} w_i \cdot \delta(a - a_i),$$

where $a_i \in \mathcal{A}(s)$, $w_i \in \mathcal{W}(s)$, and $\delta$ is the Dirac delta measure. As the number of samples goes to infinity this representation gets equivalent to the functional description of the original probability density function. This means that the actor can approximately follow the policy specified by the density $\pi^0(a|s)$, by simply choosing actions at random from $\mathcal{A}(s)$, where the (normalized) weights are the selection probabilities. Given the continuous action-value function estimated by the critic and chosen a suitable exploration strategy (e.g., the Boltzmann exploration), it is possible to define the desired probability distribution over the continuous action space, usually referred to as the *target distribution*. As long as the learning process goes on, the action values estimated by the critic become more and more reliable, and the policy followed by the agent should change in order to choose more frequently actions with higher utilities. This means that, in each state, the target distribution changes according to the information collected during the learning process, and the actor must consequently adapt its approximation.

In general, when no information is available about the shape of the target distribution, SMC methods can be effectively employed to approximate sequences of probability distributions by means of random samples, which are evolved over time exploiting *importance sampling* and *resampling* techniques. The idea behind importance sampling is to modify the weights of the samples to account for the differences between the target distribution $p(x)$ and the proposal distribution $\pi(x)$ used to generate the samples. By setting each weight $w_i$ proportional to the ratio $p(x_i)/\pi(x_i)$, the discrete weighted distribution $\sum_{i=1}^{N} w_i \cdot \delta(x - x_i)$ better approximates the target distribution. In our context, the importance sampling step is performed by the actor, which modifies the weights of the actions according to their utility values estimated by the critic. When some samples have very small or very large normalized weights, it follows that the target density significantly differs from the proposal density used to draw the samples. From a learning perspective, this means that the set of available

---

**Algorithm 1** SMC-learning algorithm

---
**for all** $s \in \mathcal{S}$ **do**
   Initialize $\mathcal{A}(s)$ by drawing $N$ samples from $\pi^0(a|s)$
   Initialize $\mathcal{W}(s)$ with uniform values: $w_i = 1/N$
**end for**
**for** each time step $t$ **do**
   *Action Selection*
   Given the current state $s_t$, the actor selects action $a_t$ from $\mathcal{A}(s_t)$ according to $\pi^t(a|s) = \sum_{i=1}^{N} w_i \cdot \delta(a - a_i)$
   *Critic Update*
   Given the reward $r_t$ and the utility of next state $s_{t+1}$, the critic updates the action value $Q(s_t, a_t)$
   *Actor Update*
   Given the action-value function, the actor updates the importance weights
   **if** the weights have a high variance **then**
     the set $\mathcal{A}(s_t)$ is resampled
   **end if**
**end for**

---

actions contains a number of samples whose estimated utility is very low. To avoid this, the actor has to modify the set of available actions by resampling new actions from the current weighted approximation of the target distribution.

In SMC-learning, SMC methods are included into a learning algorithm that iterates through three main steps (see Algorithm 1): the action selection performed by the actor, the update of the action-value function managed by the critic, and finally the update of the policy of the actor.

## 3.1 Action Selection

One of the main issues of learning in continuous action spaces is to determine which is the best action in the current state, given the (approximated) action-value function. Actor-critic methods effectively solve this problem by explicitly storing the current policy. As previously described, in SMC-learning the actor performs the action selection step by taking one action at random among those available in the current state. The probability of extraction of each action is equal to its normalized weight $Pr(a_i|s) = w_i$. The time complexity of the action selection phase for SMC-learning is logarithmic in the number of actions samples.

## 3.2 Critic Update

While the actor determines the policy, the critic, on the basis of the collected rewards, computes an approximation of the action-value function. Although several function approximation schemes could be adopted for this task (e.g., neural networks, regression tress, support-vector machines), we use a simple solution: the critic stores an action value, $Q(s, a_i)$, for each action available in state $s$ (like in tabular approaches) and modifies it according to TD update rules (see Equation 1). Using on-policy algorithms, such as SARSA, the time complexity of the critic update is constant (i.e., does not depend on the number of available actions).

## 3.3 Actor Update

The core of SMC-learning is represented by the update of the policy distribution performed by the actor. Using the importance sampling principle, the actor modifies the weights $w_i$, thus performing a policy improvement step based on the action values computed by the critic. In this way, actions with higher estimates get more weight. Several RL schemes could be adopted to update the weights. In this paper, we focus on the Boltzmann exploration strategy [13].

The *Boltzmann exploration* strategy privileges the execution of actions with higher estimated utility values. The probabilities computed by the Boltzmann exploration can be used as weights for the available actions. At time instant $t$, the weight of action $a_i$ in state $s$ is updated as follows:

$$w_i^{t+1} = w_i^t \frac{e^{\frac{\Delta Q^{t+1}(s,a_i)}{\tau}}}{\sum_{j=1}^{N} w_j e^{\frac{\Delta Q^{t+1}(s,a_j)}{\tau}}}, \tag{2}$$

where $\Delta Q^{t+1}(s, a_i) = Q^{t+1}(s, a_i) - Q^t(s, a_i)$, and the parameter $\tau$ (usually referred as to temperature) specifies the exploration degree: the higher $\tau$, the higher the exploration.

Once the weights have been modified, the agent's policy has changed. Unfortunately, it is not possible to optimally solve continuous action MDPs by exploring only a finite set of actions sampled from a prior distribution, since the optimal action may not be available. Since the prior distribution used to initialize the set of available actions significantly differs from the optimal policy distribution, after a few iterations, several actions will have negligible weights: this problem is known as the *weight degeneracy* phenomenon [1]. Since the number of samples should be kept low for efficiency reasons, having actions associated with very small weights means to waste learning parameters for approximating both the policy and the value function in regions of the action space that are not relevant with respect to the optimal policy. Furthermore, long learning time is spent to execute and update utility values of actions that are not likely to be optimal. Therefore, following the SMC approach, after the importance sampling phase, a resampling step may be needed in order to improve the distribution of the samples on the action domain. The degeneracy phenomenon can be measured through the effective sample size [8], which, for each state $s$, can be estimated by

$$\widehat{N}_{eff}(s) = \frac{1}{\displaystyle\sum_{w_i \in \mathcal{W}(s)} w_i^2},$$  (3)

where $w_i$ is the normalized weight. $\widehat{N}_{eff}(s)$ is always less than the number of actions contained in $\mathcal{A}(s)$, and low values of $\widehat{N}_{eff}(s)$ reveal high degeneracy. In order to avoid high degeneracy, the actions are resampled whenever the ratio between the effective sample size $\widehat{N}_{eff}(s)$ and the number of samples $N$ falls below some given threshold $\sigma$. The goal of resampling methods is to replace samples with small weights, with new samples close to samples with large weights, so that the discrepancy between the resampled weights is reduced. The new set of samples is generated by resampling (with replacement) $N$ times from the following discrete distribution

$$\pi(a|s) = \sum_{i=1}^{N} w_i \cdot \delta(a - a_i),$$  (4)

so that samples with high weights are selected many times. Among the several resampling approaches that have been proposed, here we consider the systematic resampling scheme, since it can be easily implemented, takes O($N$) time, and minimizes the Monte Carlo variance (refer to [1] for more details). The new samples inherit the same action values of their parents, and the sample weights are initialized using the Boltzmann distribution.

Although the resampling step reduces the degeneracy, it introduces another problem known as *sample impoverishment*. Since samples with large weights are replicated several times, after a few resampling steps a significant number of samples could be identical. Furthermore, we need to learn over a continuous space, and this cannot be carried out using a discrete set of fixed samples; in fact, the learning agent would not be able to achieve the optimal policy whenever the initial set of available actions in state $s$ ($\mathcal{A}(s)$) does not contain the optimal action of that state. This limitation may be overcome by means of a *smoothing* step, that consists of moving the samples according to a continuous approximation $\pi'(a|s, w_i)$ of the posterior distribution . The approximation is obtained by using a weighted mean of kernel densities:

$$\pi'(a|s, w_i) = \frac{1}{h} \sum_{i=1}^{N} w_i K\left(\frac{a - a_i}{h}\right),$$  (5)

where $h > 0$ is the kernel bandwidth. Typical choices for the kernel densities are Gaussian kernels and Epanechnikov kernels. However, these kernels produce over-dispersed posterior distributions, and this negatively affects the convergence speed of the learning process, especially when a few samples are used. Here, we propose to use uniform kernels:

$$K_i(a) = U\left[\frac{(a_{i-1} - a_i)}{2}; \frac{(a_{i+1} - a_i)}{2}\right].$$  (6)

As far as boundary samples are concerned (i.e., $a_1$ and $a_N$), their corresponding kernel is set to $K_1(a) = U[(a_1 - a_2); (a_2 - a_1)/2]$ and $K_N(a) = U[(a_{N-1} - a_N)/2; (a_N - a_{N-1})]$ respectively, thus preserving the possibility to cover the whole action domain. Using these (non-overlapped) kernel densities, each sample is moved locally within an interval which is determined by its distances from the adjacent samples, thus achieving fast convergence.

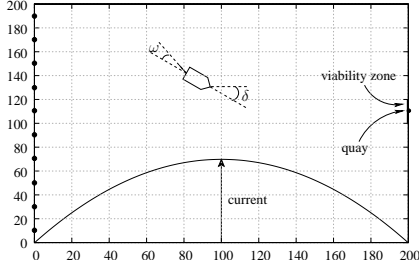

Figure 1: The boat problem.

| Parameter | Value |
|---|---|
| $f_c$ | 1.25 |
| $I$ | 0.1 |
| $s_{MAX}$ | 2.5 |
| $s_D$ | 1.75 |
| $p$ | 0.9 |
| $quay$ | (200, 110) |
| $Z_s$ width | 0.2 |
| $Z_v$ width | 20 |

Table 1: The dynamics parameters.

| Alg. | Param. | Value |
|---|---|---|
| All | $\alpha_0/\delta_\alpha$ | 0.5/0.01 |
| All | $\gamma$ | 0.99 |
| SARSA | $\tau_0/\delta_\tau$ | 3.0/0.0001 |
| SMC | $\sigma$ | 0.95 |
| SMC | $\tau_0/\delta_\tau$ | 25.0/0.0005 |
| Cont.-QL | $\epsilon/\delta_\epsilon$ | 0.4/0.005 |

Table 2: The learning parameters.

Besides reducing the dispersion of the samples, this resampling scheme implements, from the critic perspective, a variable resolution generalization approach. Since the resampled actions inherit the action value associated to their parent, the learned values are generalized over a region whose width depends on the distance between samples. As a result, at the beginning of the learning process, when the actions are approximately uniformly distributed, SMC-learning performs broad generalization, thus boosting the performance. On the other hand, when the learning is near convergence the available actions tend to group around the optimal action, thus automatically reducing generalization which may prevent the learning of the optimal policy (see [12]).

## 4 Experiments

In this section, we show experimental results with the aim of analyzing the properties of SMC-learning and to compare its performance with other RL approaches. Additional experiments on a mini-golf task and on the swing-up pendulum problem are reported in Appendix.

### 4.1 The Boat Problem

To illustrate how the SMC-learning algorithm works and to assess its effectiveness with respect to approaches based on discretization, we used a variant of the boat problem introduced in [5]. The problem is to learn a controller to drive a boat from the left bank to the right bank quay of a river, with a strong non-linear current (see Figure 1). The boat's bow coordinates, $x$ and $y$, are defined in the range $[0, 200]$ and the controller sets the desired direction $U$ over the range $[-90°, 90°]$. The dynamics of the boat's bow coordinates is described by the following equations:

$$
\begin{aligned}
x_{t+1} &= \min(200, \max(0, x_t + s_{t+1}\cos(\delta_{t+1}))) \\
y_{t+1} &= \min(200, \max(0, y_t - s_{t-1}\sin(\delta_{t+1}) - E(x_{t+1})))
\end{aligned}
$$

where the effect of the current is defined by $E(x) = f_c\left(\frac{x}{50} - \left(\frac{x}{100}\right)^2\right)$, where $f_c$ is the force of the current, and the boat angle $\delta_t$ and speed $s_t$ are updated according to the desired direction $U_{t+1}$ as:

$$
\begin{aligned}
\delta_{t+1} &= \delta_t + I\Omega_{t+1} \\
\Omega_{t+1} &= \Omega_t + ((\omega_{t+1} - \Omega_t)(s_{t+1}/s_{MAX})) \\
s_{t+1} &= s_t + (s_D - s_t)I \\
\omega_{t+1} &= \min(\max(p(U_{t+1} - \delta_t), -45°), 45°)
\end{aligned}
$$

where $I$ is the system inertia, $s_{MAX}$ is the maximum speed allowed for the boat, $s_D$ is the speed goal, $\omega$ is the rudder angle, and $p$ is a proportional coefficient used to compute the rudder angle in order to reach the desired direction $U_t$.

The reward function is defined on three bank zones. The success zone $\mathcal{Z}_s$ corresponds to the quay, the viability zone $\mathcal{Z}_v$ is defined around the quay, and the failure zone $\mathcal{Z}_f$ in all the other bank points. Therefore, the reward function is defined as:

$$
\mathcal{R}(x, y) = \begin{cases} +10 & (x, y) \in \mathcal{Z}_s \\ D(x,y) & (x, y) \in \mathcal{Z}_v \\ -10 & (x, y) \in \mathcal{Z}_f \\ 0 & \text{otherwise} \end{cases} \tag{7}
$$

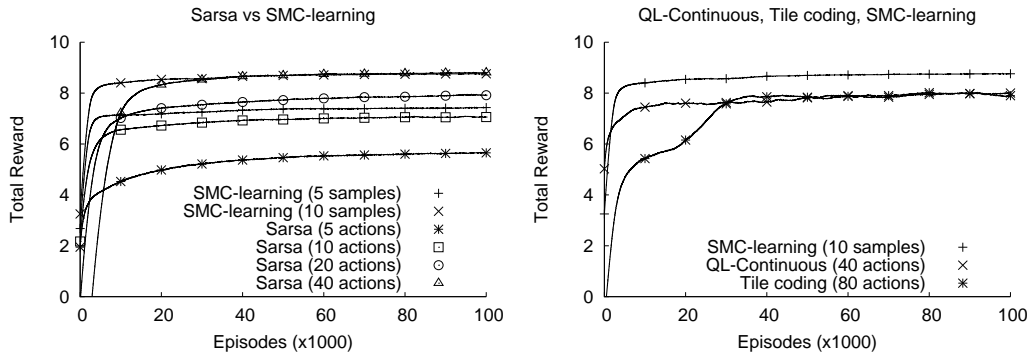

Figure 2: Performance comparison between SMC-learning and SARSA *(left)*, SMC-learning and tile coding and Continuous Q-learning *(right)*

where $D$ is a function that gives a reward decreasing linearly from 10 to -10 relative to the distance from the success zone. In the experiment, each state variable is discretized in 10 intervals and the parameters of the dynamics are those listed in Table 1. At each trial, the boat is positioned at random along the left bank in one of the points shown in Figure 1. In the following, we compare the results obtained with four different algorithms: SARSA with Boltzmann exploration with different discretizations of the action space, SARSA with tile coding (or CMAC) [12], Continuous Q-learning [9], and SMC-learning. The learning parameters of each algorithm are listed in Table 2. [1]

Figure 2-*left* compares the learning performance (in terms of total reward per episode) for SARSA with 5, 10, 20, and 40 evenly distributed actions to the results obtained by SMC-learning with 5 and 10 samples. As it can be noticed, the more the number of actions available the better the performance of SARSA is. With only 5 actions (one action each $36°$), the paths that the controller can follow are quite limited and the quay is not reachable from any of the starting point. As a result, the controller learned by SARSA achieves a very poor performance. On the other hand, a finer discretization allows the boat to reach more frequently the quay, even if it takes about three times the number of episodes to converge with respect to the case with 5 actions. As it can be noticed, SMC-learning with 5 samples outperforms SARSA with 5 and 10 actions both in terms of performance and in convergence time. In fact, after few trials, SMC-learning succeeds to remove the less-valued samples and to add new samples in regions of the action space where higher rewards can be obtained. As a result, not only it can achieve better performance than SARSA, but it does not spend time exploring useless actions, thus improving also the convergence time. Nonetheless, with only 5 samples the actor stores a very roughly approximated policy, which, as a consequence of resampling, may converge to actions that do not obtain a performance as good as that of SARSA with 20 and 40 actions. By increasing the number of samples from 5 to 10, SMC-learning succeeds in realizing a better coverage of the action space, and obtains equivalent performance as SARSA with 40 actions. At the same time, while the more actions available, the more SARSA takes to converge, the convergence time of SMC-learning, as in the case with 5 samples, benefits from the initial resampling, thus taking less than one sixth of the trials needed by SARSA to converge.

Figure 2-*right* shows the comparison of the performance of SMC-learning, SARSA with tile coding using two tilings and a resolution of $2.25°$ (equivalent to 80 actions), and Continuous Q-learning with 40 actions. We omit the results with fewer actions because both tile coding and Continuous Q-learning obtain poor performance. As it can be noticed, SMC-learning outperforms both the compared algorithms. In particular, the generalization over the action space performed by tile coding negatively affects the learning performance because of the non-linearity of the dynamics of the system. In fact, when only few actions are available, two adjacent actions may have completely different effects on the dynamics and, thus, receive different rewards. Generalizing over these actions prevents the agent from learning which is the best action among those available. On the other hand, as long as the samples get closer, SMC-learning dynamically reduces its generalization over the ac-

tion space, so that their utility can be more accurately estimated. Similarly, Continuous Q-learning is strictly related to the actions provided by the designer and to the implicit assumption of linearity of the action-value function. As a result, although it could learn any real-valued action, it does not succeed in obtaining the same performance as SMC-learning even with the quadruple of actions. In fact, the capability of SMC-learning to move samples towards more rewarding regions of the action space allows the agent to learn more effective policies even with a very limited number of samples.

## 5 Conclusions

In this paper, we have described a novel actor-critic algorithm to solve continuous action problems. The algorithm is based on a Sequential Monte Carlo approach that allows the actor to represent the current policy through a finite set of available actions associated to weights, which are updated using the utility values computed by the critic. Experimental results show that SMC-learning is able to identify the highest valued actions through a process of importance sampling and resampling. This allows SMC-learning to obtain better performance with respect to static solutions such as Continuous Q-learning and tile coding even with a very limited number of samples, thus improving also the convergence time. Future research activity will follow two main directions: extending SMC-learning to problems in which no good discretization of the state space is a priori known, and experimenting in continuous action multi-agent problems.

## Footnotes

[1] $\delta_x$ is the decreasing rate for parameter $x$, whose value after $N$ trials is computed as $x(N) = \frac{x(0)}{1+\delta_x N}$.

## References

[1] M. Sanjeev Arulampalam, Simon Maskell, Neil Gordon, and Tim Clapp. A tutorial on particle filters for online nonlinear/non-gaussian bayesian tracking. *IEEE Trans. on Signal Processing*, 50(2):174–188, 2002.

[2] Leemon C. Baird and A. Harry Klopf. Reinforcement learning with high-dimensional, continuous actions. Technical Report WL-TR-93-117, Wright-Patterson Air Force Base Ohio: Wright Laboratory, 1993.

[3] D.P. Bertsekas and J.N. Tsitsiklis. *Neural Dynamic Programming*. Athena Scientific, Belmont, MA, 1996.

[4] Chris Gaskett, David Wettergreen, and Alexander Zelinsky. Q-learning in continuous state and action spaces. In *Australian Joint Conference on Artificial Intelligence*, pages 417–428, 2003.

[5] L. Jouffe. Fuzzy inference system learning by reinforcement methods. *IEEE Trans. on Systems, Man, and Cybernetics-PART C*, 28(3):338–355, 1998.

[6] H. Kimura and S. Kobayashi. Reinforcement learning for continuous action using stochastic gradient ascent. In *5th Intl. Conf. on Intelligent Autonomous Systems*, pages 288–295, 1998.

[7] V. R. Konda and J. N. Tsitsiklis. Actor-critic algorithms. *SIAM Journal on Control and Optimization*, 42(4):1143–1166, 2003.

[8] J. S. Liu and E. Chen. Sequential monte carlo methods for dynamical systems. *Journal of American Statistical Association*, 93:1032–1044, 1998.

[9] Jose Del R. Millan, Daniele Posenato, and Eric Dedieu. Continuous-action q-learning. *Machine Learning*, 49:247–265, 2002.

[10] Jan Peters and Stefen Schaal. Policy gradient methods for robotics. In *Proceedings of the IEEE International Conference on Intelligent Robotics Systems (IROS)*, pages 2219–2225, 2006.

[11] J. C. Santamaria, R. S: Sutton, and A. Ram. Experiments with reinforcement learning in problems with continuous state and action spaces. *Adaptive Behavior*, 6:163–217, 1998.

[12] Alexander A. Sherstov and Peter Stone. Function approximation via tile coding: Automating parameter choice. In *SARA 2005*, LNAI, pages 194–205. Springer Verlag, 2005.

[13] Richard S. Sutton and Andrew G. Barto. *Reinforcement Learning: An Introduction*. MIT Press, Cambridge, MA, 1998.

[14] Hado van Hasselt and Marco Wiering. Reinforcement learning in continuous action spaces. In *2007 IEEE Symposium on Approximate Dynamic Programming and Reinforcement Learning*, pages 272–279, 2007.
